# Strategic Impatience in Go/NoGo versus Forced-Choice Decision-Making

**Pradeep Shenoy**
Cognitive Science Department
University of California, San Diego
La Jolla, CA, 92093
pshenoy@ucsd.edu

**Angela J. Yu**
Cognitive Science Department
University of California, San Diego
La Jolla, CA, 92093
ajyu@ucsd.edu

## Abstract

Two-alternative forced choice (2AFC) and Go/NoGo (GNG) tasks are behavioral choice paradigms commonly used to study sensory and cognitive processing in choice behavior. While GNG is thought to isolate the sensory/decisional component by eliminating the need for response selection as in 2AFC, a consistent tendency for subjects to make more Go responses (both higher hits and false alarm rates) in the GNG task raises the concern that there may be fundamental differences in the sensory or cognitive processes engaged in the two tasks. Existing mechanistic models of these choice tasks, mostly variants of the drift-diffusion model (DDM; [1, 2]) and the related leaky competing accumulator models [3, 4], capture various aspects of behavioral performance, but do not clarify the provenance of the Go bias in GNG. We postulate that this "impatience" to go is a strategic adjustment in response to the implicit asymmetry in the cost structure of the 2AFC and GNG tasks: the NoGo response requires waiting until the response deadline, while a Go response immediately terminates the current trial. We show that a Bayes-risk minimizing decision policy that minimizes not only error rate but also average decision delay naturally exhibits the experimentally observed Go bias. The optimal decision policy is formally equivalent to a DDM with a *time-varying threshold* that initially rises after stimulus onset, and collapses again just before the response deadline. The initial rise in the threshold is due to the diminishing temporal advantage of choosing the fast Go response compared to the fixed-delay NoGo response. We also show that fitting a simpler, fixed-threshold DDM to the optimal model reproduces the counterintuitive result of a higher threshold in GNG than 2AFC decision-making, previously observed in direct DDM fit to behavioral data [2], although such fixed-threshold approximations cannot reproduce the Go bias. Our results suggest that observed discrepancies between GNG and 2AFC decision-making may arise from rational strategic adjustments to the cost structure, and thus need not imply any other difference in the underlying sensory and cognitive processes.

## 1 Introduction

The two-alternative forced-choice (2AFC) task is a standard experimental paradigm used in psychology and neuroscience to investigate various aspects of sensory, motor, and cognitive processing [5]. Typically, the paradigm involves a forced choice between two responses based on a presented stimulus, with the measured response time and accuracy of choices shedding light on the cognitive and neural processes underlying behavior. Another paradigm that appears to share many features of the 2AFC task is the Go/NoGo (GNG) task [6], (see Luce [5] for a review), where one stimulus category is associated with an overt *Go* response that has to be executed before a response dead-

line, and the other stimulus (NoGo) requires withholding response until the response deadline has elapsed. In principle, the GNG task could be used to probe the same decision-making problems as the 2AFC task, with the possible advantage of eliminating a "response selection stage" that may follow the decision in the 2AFC task [6, 7]. Indeed, the GNG task has been used to study various aspects of human and animal cognition, e.g., lexical judgements [8, 9], perceptual decision-making [10, 11, 12], and the neural basis of choice behavior (in particular, distinguishing among neural activations associated with stimulus, memory, and response) [13, 14, 15]. However, experimental evidence also indicates that there is a curious choice bias toward the overt (Go) response in the GNG task [11, 16, 2, 15], in the form of shorter response times and more false alarms for the Go response, than when compared to the same stimulus pairings in a 2AFC task [2, 16]. It has been suggested that this choice bias may reflect differential sensory and cognitive processes underlying the two tasks, and thus making the two non-interchangeable in the study of perception and decision-making.

In this paper, we hypothesize that this discrepancy may simply be due to differences in the implicit reward (cost) structure of the two tasks: the NoGo response incurs a higher imposed waiting cost than the Go response, since the NoGo response must wait until the response deadline has passed to register, while a Go response immediately terminates the trial. In contrast, in the 2AFC task, the cost function is symmetric for the two alternatives, whether in terms of error or delay. We propose that the implicit cost structure difference in GNG can fully account for the Go bias in GNG compared to 2AFC tasks, without the need to appeal to other differences in sensory or cognitive processing. To investigate this hypothesis, we adopt a Bayes risk minimization framework for both the 2AFC and GNG tasks, whereby sensory processing is modeled as iterative Bayesian inference of stimulus type based on a stream of noisy sensory input, and the decision of when/how to respond rests on a policy that minimizes a linear combination of expected decision delay and response errors. The optimal decision policy for this Bayes-risk formulation in the 2AFC task is known as the sequential probability ratio test (SPRT; [17, 18]), and has been shown to account for both behavioral [19, 4] and neural data [19, 20]. Here, we generalize this theoretical framework to account for both 2AFC and GNG decision-making in a unified framework, by assuming that a subject's sensory and perceptual processing (of the same pair of stimuli) and the relative preference for decision accuracy versus speed are *shared* across 2AFC and GNG, with the only difference between them being the asymmetric temporal cost implicit in the reward structure of the GNG task –the Go response terminating a trial while the NoGO response only registering after the response deadline.

As a stochastic process, SPRT is a bounded random walk, whereby the stochasticity in the random walk comes from noise in the observation process. The continuum (time) limit of a bounded random walk is the bounded drift-diffusion model (DDM), which generally assume a stochastic dynamic variable to undergo constant drift, as well as diffusion due to Wiener noise, until one of two finite thresholds is breached. In psychology, DDM has been augmented with additional parameters such as a non-decision-related repsonse delay, variability in drift-rate, and variability in starting point across trials. Figure 4A shows a simple variant of the DDM illustrating the following parameters: rate of accumulation, threshold, and "nondecision time" or temporal offset to the start of the diffusion process. These augmented DDMs have been used to model behavior in 2AFC tasks [21, 22, 23, 5, 24, 4], and also appear to provide good descriptive accounts of the neural activities underlying perceptual decision-making [25, 20, 26, 27]. Variants of augmented DDM have also been utilized to fit data in other simple decision-making tasks, including the GNG task [2]. While augmenting DDM with extra parameters gives it additional power in explaining subtleties in data, this also diminishes the normative interpretability of DDM fits by eliminating its formal relationship to the optimal SPRT procedure. As a consequence, when the behavioral objectives change, e.g., in the GNG task, DDM cannot predict *a priori* what parameters ought to change and how much. Instead, we begin with a Bayes-risk minimization formulation and derive the non-parametric optimal decision-procedure as a function of sensory statistics and behavioral objectives. We then map the optimal policy to the DDM model space, and compare directly with previously proposed DDM variants in the context of 2AFC and GNG tasks.

In the following sections, we first describe our proposed Bayesian inference and decision-making model, then compare simulations of the optimal decision-making model with published experimental data of subjects performing perceptual decision-making in 2AFC and GNG tasks [16]. We also explore other evidence exploring the degree of go bias in the GNG task [28]. Next, we consider the formal relationship between the optimal model and a fixed-threshold DDM that was previously utilized to fit behavioral data from the GNG task [2, 12]. Finally, we present novel experimental

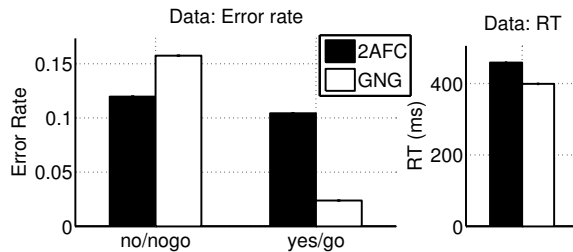

Figure 1: Systematic error biases in the GNG task. (A) The figure shows error rates associated with a perceptual decision-making task performed by subjects in both Go/NoGo and Yes/No (forced choice) settings. Although the error rates in the forced choice settings were similar for both classes, there was a significant bias towards the Go response in the GNG task, with more false alarms than omission errors. (B) Mean response time on the GNG task was lower than for the same stimulus on the 2AFC task. (Data adapted from Bacon-Mace et al., 2007).

predictions of the optimal decision-making model, including those that specifically differ from the fixed-threshold DDM approximation [2, 12].

## 2   Bayesian inference and risk minimization in choice tasks

Human choice behavior in the GNG and 2AFC tasks exhibits a consistent Go bias in the GNG task that is not apparent for the same stimulus in the 2AFC task. For example, Figure 1 shows data from a task in which subjects must identify whether a briefly-presented noisy image contains an animal or not [16], under two different response conditions: GNG (only respond to animal-present images), and 2AFC (respond yes/no to each image). Subjects showed a significant bias towards the Go response in the GNG task, in the form of higher false alarms than omission errors (Figure 1A), as well as faster RT than for the same stimulus in the 2AFC task (Figure 1B).

For the 2AFC task, a large body of literature supports the "accumulate-to-bound" model of perceptual decision-making, [23, 20, 26], where moment-to-moment sensory input ("evidence" in favor of either choice) is accumulated over time until it reaches a *bound*, at which point, a response is generated. Previous work by Yu & Frazier [29] extended the formulation to include 2AFC tasks with a decision deadline, in which subjects have the additional constraint of not exceeding a decision deadline. They showed that the optimal policy for decision-making under a deadline is to accumulate evidence up to *time-varying thresholds* that collapse toward each other over time, leading to more "liberal" choices and higher error rate in later responses than earlier ones. Here, we generalize the framework to model the GNG task. In particular, the same deadline by which the subject must make a response (or else be counted as a "miss") on a Go trial, is the one for which the subject must withold response (or else be counted as a "false alarm"). We model evidence accumulation as iterative Bayesian inference over the identity of the stimulus, and decision-making as an iterative decision policy that chooses whether to respond (and which one in 2AFC) or continue observing at least one more time point, based on current evidence. The optimal policy minimizes the expected value of a cost function that depends linearly on decision delay and errors. The model is described below.

### 2.1   Evidence integration as Bayesian inference

We model evidence accumulation, in both 2AFC and GNG, as iterative Bayesian inference about the stimulus identity conditioned on an independent and identically distributed (i.i.d.) stream of sensory input. Specifically, we assume a *generative model* where the observations are a continual sequence of data samples $x_1, x_2, \ldots$, iid-generated from a likelihood function $f_0(x)$ or $f_1(x)$ depending on whether the true stimulus state is $d = 0$ or $d = 1$, respectively. This incoming stream therefore provides accumulating evidence of the hidden category label $d \in \{0, 1\}$. For concreteness, we assume the likelihood functions are Gaussian distributions with means $\pm\mu$ (+ for $d = 1$, - for $d = 0$), and a variance parameter $\sigma^2$ controlling the noisiness of the stimuli.

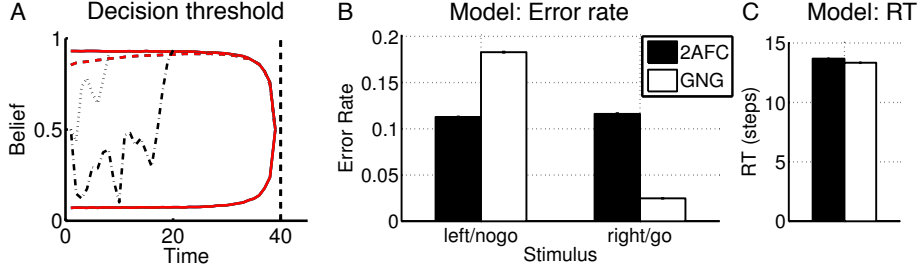

Figure 2: Rational behavior in 2AFC and GNG tasks. (A) The figure shows the decision threshold as a function of belief state across the 2AFC and GNG tasks. The optimal decision boundary for 2AFC is a pair of parallel thresholds (solid line) that collapse and meet at the response deadline (indicated by dashed vertical line). The optimal GNG decision boundary is a single initially *increasing* threshold (dashed line), that decreases to 0.5 at the response deadline. (B;C) Monte Carlo simulation of the optimal policy show a bias towards the overt response in the GNG task. The two response alternatives in the 2AFC task are represented as "left" and "right", corresponding to "nogo" and "go" in the GNG task (B). The GNG task shows lower miss rate and higher false alarm rate than the corresponding 2AFC error rate (B), along with faster RT than the 2AFC task (C). Compare to the experimental data in Figure 1. Parameter settings: $c = 0.01, \mu = 0.25, D = 40$ timesteps.

The *recognition model* specifies the mechanism by which stimulus identity is *inferred* from the noisy observations $\mathbf{x}_t$. In our model, we compute an posterior distribution over the category label conditioned on the data sampled so far $\mathbf{x}_t \triangleq (x_1, x_2, \ldots x_t)$, $b_t \triangleq P\{d = 1 | \mathbf{x}_t\}$, also known as the *belief* state, by iteratively applying Bayes' rule:

$$b_{t+1} = \frac{b_t f_1(x_{t+1})}{b_t f_1(x_{t+1}) + (1 - b_t) f_0(x_{t+1})} \tag{1}$$

where $b_0 \triangleq P\{d = 1\}$ is the prior probability of the stimulus category being 1 (and is 0.5 for equally likely stimuli). We hypothesize that the same evidence accumulation mechanism underlies decision-making in both tasks, in particular with the same noise process/likelihood functions, $f_0(x)$ and $f_1(x)$, for a particular individual observing the same stimuli.

## 2.2 Action selection as Bayes-risk minimization

We model behavior in the two tasks as a sequential decision-making process where, at each instant, the model decideses between two actions, as a function of the current evidence so far, encapsulated in the current belief state $b_t$: *stop* (and choose the response for the more probable stimulus category for 2AFC), or *continue* one more time step. A stopping policy is a mapping from the belief state to the action space, $\pi :b_t \mapsto \{stop, continue\}$, where the *stop* action in 2AFC also requires a stimulus category decision $\delta$. In accordance with the standard Bayes risk framework for optimizing the decision policy in a stopping problem, we assume that the behavioral cost function is a linear combination of the probability of making a decision error and the expected decision delay $\tau$ (the stopping time if a response is emitted before the deadline, and the deadline $D$ otherwise). We assume that the decision delay component is weighted by a *sampling* or *time* cost $c$, while the cost of all decision errors are penalized by the same magnitude and normalized to unit cost. Based on this cost function, the optimal decision policy is the policy that minimizes the overall *expected cost*:

$$2AFC : L_\pi = c\langle\tau\rangle + P\{\delta \neq d\} + P\{\tau = D\} \tag{2}$$
$$GNG : L_\pi = c\langle\tau\rangle + P\{\tau = D | d = 1\}P\{d = 1\} + P\{\tau < D | d = 0\}P\{d = 0\} \tag{3}$$

The 2AFC cost function is a special case of the more general scenario previously considered for deadlined sequential hypothesis testing [29]: $P\{\delta \neq d\}$ is the expected wrong response cost, while $P\{\tau = D\}$ is the expected cost of not responding before the deadline (omission error). In the GNG cost function, $P\{\tau = D | d = 1\}$ is the probability that no response is emitted before the deadline on a Go trial (miss), $P\{\tau < D | d = 0\}$ is the probability that a NoGo trial is terminated by a Go

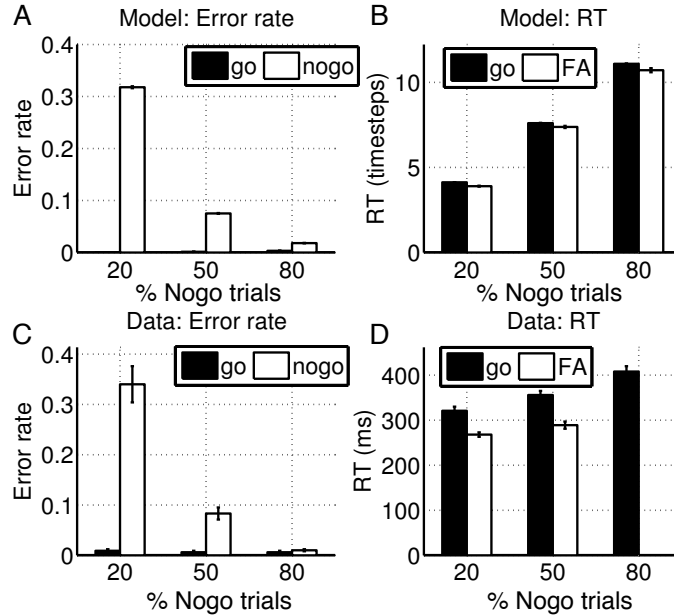

Figure 3: Influence of stimulus statistics on Go bias. Our model predicts that false alarms are more frequent than misses (A), and are also faster than correct Go RTs (B). The Go bias, which is apparent at 50% Go trials, is signficantly increased when Go trials are more frequent (80%), and reduced when Go trials are reduced to 20% of the trials. Parameter settings: $c = 0.014, \mu = 0.45, D = 40$ timesteps. (C-D) Human subjects exhibited a similar pattern of behavior in a letter discrimination task (Data from Nieuwenhuis et al., 2003).

response (false alarm), a correct hit requires $\tau < D$ (responding before the deadline), and a correct NoGo response consists of a series of *continue* actions until a predefined response deadline $D$. In both GNG and 2AFC tasks, the choice to *stop* limits the decision delay cost, and the choice to *continue* (up to a predefined response deadline $D$) results in the collection of more data that help to disambiguate the stimulus category but at the cost of $c$ per additional sample of data observed. We compute the optimal policy using Bellman's dynamic programming principle (Bellman, 1952). Specifically, we iteratively compute the expected cost of *continue* and *stop* as a function of the belief state $b_t$ (these are the *Q-factors* for *continue* and *stop*, $Q_c(b_t)$ and $Q_s(b_t)$). If $Q_c(b_t) < Q_s(b_t)$, then the optimal policy chooses to *continue*; otherwise, it chooses to stop; therefore, the belief state is partitioned by the decision policy into a *continuation region* and a *stopping region* (details omitted due to lack of space).

The principal difference between the two tasks as formulated here is the loss function. In the 2AFC task, all trials are terminated by a response (unless the response deadline is exceeded). However, in the GNG version, subjects have to wait until the response deadline to choose the NoGo response. This introduces a significant, extra cost of time for NoGo responses, suggesting that it may in some cases be better to select the Go response despite the relative inadequacy of sensory evidence. We explore these aspects in detail in the following section.

## 3 Results

**Opportunity cost and the Go/NoGo decision threshold**
Figure 2A illustrates the difference between the optimal decision policies for the two tasks. The red lines (solid: 2AFC, dashed: GNG) illustrates the optimal decision thresholds, which, when exceeded by the cumulative sensory evidence $b_t$, generate the corresponding response, as a function of time. For the 2AFC task, the optimal policy is a pair of thresholds that are initially fairly constant over time, but then collapse toward each other (into an empty set if the cost of exceeding the deadline is sufficiently large) as the deadline approaches (cf. [29]). In contrast, the threshold for the GNG

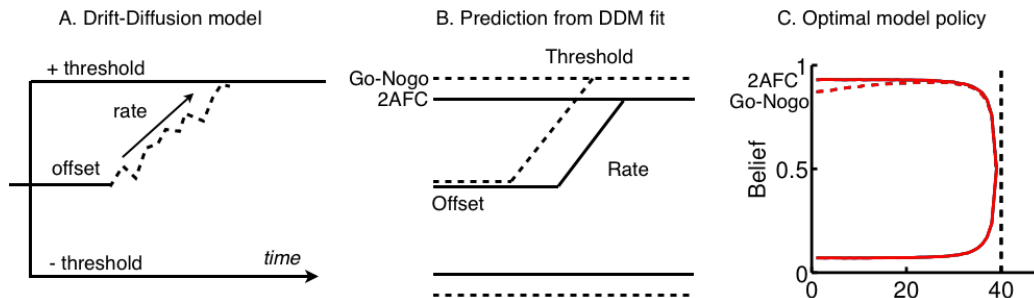

Figure 4: Drift-diffusion model (DDM) for 2AFC and GNG tasks (A) A simplified version of the DDM for 2-choice tasks, where a noisy accumulation process with a certain *rate* produces one of two responses when it reaches a positive or negative *threshold*. In addition to the rate and threshold parameters, a third parameter (the temporal *offset* to the start of the accumulation process) represents the nondecision processes associated with visual and motor delays. (B) DDM fits to 2AFC and GNG choice data(Gomez et al., 2007, Mack & Palmeri, 2010) suggest that the GNG task is associated with a higher threshold and shorter offset than the 2AFC task. (C) Optimal decision-making model predicts a lower, time-varying threshold for the GNG task.

task (dotted line) is a single threshold that varies over time, and is *lower* at the beginning of the trial. This is a direct consequence of the opportunity cost involved with waiting until the deadline: if the deadline is far away, the cost of waiting may be more than the cost of an immediate error that terminates the trial; indeed, we expect that the farther away the deadline, the greater temporal cost savings conferred by Go response over waiting to register the NoGo response.

**Decision-making in 2AFC and GNG tasks**
Figure 2B;C shows the effect of the time-varying threshold on RT and accuracy in an example model simulation. Figure 2B shows that the GNG model is significantly biased towards the Go response, with a higher fraction of false alarms than misses. This asymmetry is absent in the 2AFC model performance. In addition, GNG response times are *faster* than 2AFC response times (Figure 2C). This bias is a direct result of the time-varying threshold in the GNG task; early on in the trial, the decision threshold is lower, and produces fast, error-prone responses.

This model prediction is consistent with data from human perceptual decision-making. Figure 1 shows behavioral data in the two tasks [16]– subjects determined from a brief presentation of a noisy visual stimulus whether or not the image contained an animal. The same task was performed in two response conditions: 2AFC, where each stimulus required a yes/no response, and GNG, where subjects only responded to image containing the target. Figure 1A shows that in the 2AFC condition, subjects are not significantly biased towards either response, with both false alarms and miss rates being similar to each other. On the other hand, in the Go/NoGo condition, subjects showed a significant bias towards the overt response, thus producing substantially more false alarms and fewer misses. In the GNG task, their RT was significantly shorter than in the 2AFC task (Figure 1B). Similar results have also been reported by Gomez et al. in the context of lexical decision-making [2].

**Influence of stimulus probability on Go bias**
We investigate the degree of Go bias in the GNG model by considering the effect of trial type frequency on behavioral measures in the GNG task. Model simulations (Figure 3) show that, consistent with Figure 2 and a host of other experimental data, there is a significant bias toward the Go response when Go and NoGo trials are equiprobable, and this bias is increased (respectively diminished) as NoGo trials are fewer or more frequent. The figure also shows that RT for both correct Go and erroneous NoGo responses increase with the frequency of NoGo trials, and that false alarm RT is faster than correct response RT. In recent work, Nieuwenhuis et al. [28] used a block design to compare choice accuracy and RT in a letter discrimination task when the fraction of NoGo trials was set to 20%, 50%, and 80%. As shown in Figure 3C;D, , subjects' behavior was reliably modulated by trial type frequency, in a manner closely reflecting model predictions.

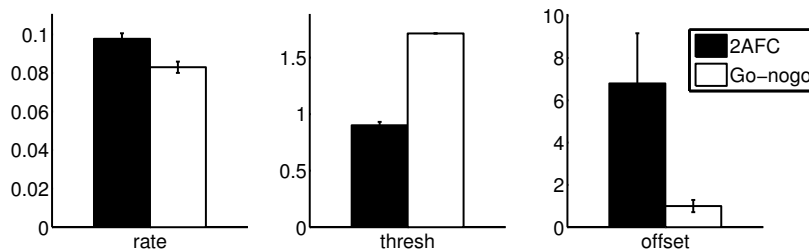

Figure 5: DDM approximation to optimal decision-making model. Simplified DDMs were fit to optimal model simulations of 2AFC and GNG behavior, and the best-fit parameters compared between tasks. The DDM approximation for optimal GNG behavior shows a higher decision threshold (B), and lower nondecision time (C), than the DDM approximation for the 2AFC task. In addition, the rate of evidence accumulation was also lower for the GNG fit (A).

In our formulation, although the decision boundary is unchanged by the experimental manipulation, the stimulus frequency induces a *prior belief* over the identity of the stimulus, and thus represents the starting point for the evidence accumulation process. When Go trials are rare, the starting point is far from the decision boundary, and it takes longer for a response to be generated. Further, due to the extra evidence needed to overcome the prior, choices are less likely to be erroneous.

**Drift-diffusion models and optimal behavior**

Various versions of augmented DDM have been used to fit GNG behavioral data, with one variant in particular suggesting that the decision threshold in GNG ought to be *higher* than 2AFC [2], in an apparent contradiction to our model's predictions (Figure 4). By fitting RT and choice data from lexical judgment, numerosity judgment, and memory-based decision making tasks, Gomez et al. [2] found that a DDM with an *implicit negative boundary* associated with the NoGo stimulus provided a good fit to RT data. Further, joint parameter fits to 2AFC and GNG choice data indicated that the principal difference in the two tasks was in the nondecision time and decision threshold; the rate parameter (representing the evidence accumulation process) was similar in both tasks. In particular, they suggested that the nondecision time was shorter, and the decision threshold higher than in the 2AFC task (Figure 4B). These results were replicated by Mack & Palmeri by fitting DDM to behavioral data from a visual categorization task performed in both 2AFC and GNG versions [12].

Although DDMs are formally equivalent to optimal decision-making in a restricted class of sequential choice problems [18], they do not explicitly represent and manipulate uncertainty and cost, as we do in our Bayesian risk-minimization framework. In particular, our framework allows us to *predict* that optimal behavior is well-characterized by a DDM with a time-varying threshold (Figure 4C), and that the restricted class of constant-threshold DDMs are insufficient to fully explain observed behavior. Nevertheless, we can ask whether our prediction is *consistent* with the empirical results obtained from DDM fits with constant decision thresholds.

To address this, we computed the best constant-threshold DDM approximations to optimal decision making in the two tasks. We simulated the optimal model with a shared set of parameters for both the 2AFC and GNG tasks, and fit simplified random-walk models with 3 free parameters (Figure 4A) to the output of our optimal model's simulations. Figure 5 shows that the best-fitting DDM approximation for optimal GNG behavior has a higher threshold and a lower offset parameter than the best-fitting DDM for optimal 2AFC task behavior.

Note that varying the magnitude of a symmetric (explicit and implicit) decision threshold is not capable of explaining the go bias towards the overt response. Gomez et al. also considered additional variants of the DDM which allow for a change in the initial starting point, and for a different accumulation rate in the GNG task. These models, when fit to data, showed a bias towards the overt response; however, the quality of fit did not significantly improve [2].

Thus, our results and those of Gomez et al. [2] are conceptually consistent; a prinicipal difference in the two tasks is the decision threshold, whereas the evidence accumulation process is similar across tasks. However, our analysis explains precisely *how* and *why* the thresholds in the two tasks are different: the GNG task has a time-varying threshold that is lower than the 2-choice threshold,

due to the difference in loss functions in the two tasks. In particular, our model accounts for the bias towards the overt response, without recourse to an implicit decision boundary or additional parameter changes. When optimal behavior is approximated by a simpler class of models (e.g., models with fixed decision threshold), the best fit to optimal GNG behavior turns out to be a higher threshold and shorter nondecision time, as found by previous work [2, 12], and adjustments to the initial starting point are required to explain the overt response bias.

## 4    Discussion

Forcing a choice between two alternatives is a fundamental technique used to study a wide variety of perceptual and cognitive phenomena, but there has long been confusion over whether GNG and 2AFC variants of such tasks are probing the same underlying neural and cognitive processes. Our work demonstrates that a common Bayes-optimal sequential inference and decision policy can explain the behavioral results in both tasks, as well as what was perceived to be a troubling Go bias in the GNG task, compared to 2AFC. We showed that the Go bias arises naturally as a rational response to the asymmetric time cost between Go and NoGo responses, as the former immediately terminates the trial, while the latter requires the subject to wait until the end of the trial to record the choice. The consequence of this cost asymmetry is an optimal decision policy that requires Bayesian evidence accumulation up to a *time-varying* boundary, which has an inverted-U shape: the initial low boundary is due to the temporal advantage of choosing to Go early and save on the time necessary to wait to register a NoGo response, the later collapsing of boundary is due to the expectation of the deadline for responding. We showed that this optimal decision policy accounts for the general behavioral phenomena observed in GNG tasks, in particular accounting for the Go bias. Importantly, our work shows that need not be any fundamental differences in the cognitive and neural processes underlying perception and decision-making in these tasks, at least not on account of the Go bias.

Our model makes several novel experimental predictions for the GNG task: (1) for fast responses, false alarm rate increases as a function of response time (in contrast, the fixed-threshold DDM approximation predicts a constant alarm rate); (2) lengthening the response deadline should exacerbate the Go bias; (3) if GNG and 2AFC share a common inference and decision-making neural infrastructure, then our model predicts within-subject cross-task correlation: e.g. favoring speed over accuracy in the 2AFC task should correlate with a greater Go bias in the GNG task.

The optimal decision policy for the GNG task can naturally be viewed as a stochastic process (though it is normatively derived from task statistics and behavioral goals). We can therefore compare our model to other stochastic process models previously proposed for the GNG task. Our model has a single decision threshold associated with the overt response, consistent with some early models proposed for the task (see e.g., Sperling et al. [30]). In contrast, the extended DDM framework proposed by Gomez et al. has an additional boundary associated with the NoGo response (corresponding to a *covert* NoGo response). Gomez et al. report that single-threshold variants of the DDM provided very poor fits to the data. Although computationally and behaviorally we do not *require* a covert-response or associated threshold, it is nevertheless possible that neural implementations of behavior in the task may involve an explicit "NoGo" choice For instance, substantial empirical work aims to isolate neural correlates of *restraint*, corresponding to a putative "NoGo" action, by contrasting neural activity on "go" and "nogo" (see e.g., [31, 32]). We will consider approximating the optimal policy with one that includes this second boundary in future work.

# References

[1] R Ratcliff and P L Smith. *Psychol. Rev.*, 111:333–346, 2004.

[2] P Gomez, R Ratcliff, and M Perea. *Journal of Experimental Psychology*, 136(3):389–413, 2007.

[3] M Usher and J L McClelland. *Psychol. Rev.*, 108(3):550–592, 2001.

[4] R. Bogacz, E. Brown, J. Moehlis, P. Holmes, and J.D. Cohen. *Psychological Review*, 113(4):700, 2006.

[5] R.D. Luce. Number 8. Oxford University Press, USA, 1991.

[6] F.C. Donders. *Acta Psychologica*, 30:412, 1969.

[7] B. Gordon and A. Caramazza. *Brain and Language*, 15(1):143–160, 1982.

[8] Y Hino and SJ Lupker. *Journal of experimental psychology. Human perception and performance*, 26:166–183, 2000.

[9] M Perea, E Rosa, and C Gomez. *Memory and Cognition*, 30(34-45), 2002.

[10] S. Thorpe, D. Fize, C. Marlot, and Others. *Nature*, 381(6582):520–522, 1996.

[11] A. Delorme, G. Richard, and M. Fabre-Thorpe. *Vision Research*, 40(16):2187–2200, 2000.

[12] ML Mack and TJ Palmeri. *Journal of Vision*, 10:1–11, 2010.

[13] M.A. Sommer and R.H. Wurtz. *J Neurophysiol.*, 85(4):1673–1685, 2001.

[14] RP Hasegawa, BW Peterson, and ME Goldberg. *Neuron*, 43(3):415–25, August 2004.

[15] G Aston-Jones, J Rajkowski, and P Kubiak. *J Neurosci.*, 14:4467–4480, 1994.

[16] N. Bacon-Macé, H. Kirchner, M. Fabre-Thorpe, and S.J. Thorpe. *J Exp. Psychol.: Human Perception and Performance*, 33(5):1013, 2007.

[17] A Wald. Dover publications, 1947.

[18] A. Wald and J. Wolfowitz. *The Annals of Mathematical Statistics*, 19(3):326–339, 1948.

[19] J.D. Roitman and M.N. Shadlen. *J neurosci.*, 22(21):9475, 2002.

[20] J.I. Gold and M.N. Shadlen. *Neuron*, 36(2):299–308, 2002.

[21] M. Stone. *Psychometrika*, 25(3):251–260, 1960.

[22] D.R.J. Laming. Academic Press, 1968.

[23] R. Ratcliff. *Psychological Review*, 85(2):59, 1978.

[24] J.I. Gold and M.N. Shadlen. *Annu. Rev. Neurosci.*, 30:535–574, 2007.

[25] D.P. Hanes and J.D. Schall. *Science*, 274(5286):427, 1996.

[26] M.E. Mazurek, J.D. Roitman, J. Ditterich, and M.N. Shadlen. *Cerebral cortex*, 13(11):1257, 2003.

[27] R Ratcliff, A Cherian, and M Segraves. *Journal of neurophysiology*, 90:1392–1407, 2003.

[28] S Nieuwenhuis, N Yeung, W van den Wildenberg, and KR Ridderinkhof. *Cognitive, affective & behavioral neuroscience*, 3(1):17–26, March 2003.

[29] P. Frazier and A.J. Yu. *Advances in neural information processing systems*, 20:465–472, 2008.

[30] G. Sperling and B. Dosher. *Handbook of perception and human performance.*, 1:2–1, 1986.

[31] D.J. Simmonds, J.J. Pekar, and S.H. Mostofsky. *Neuropsychologia*, 46(1):224–232, 2008.

[32] A.R. Aron, S. Durston, D.M. Eagle, G.D. Logan, C.M. Stinear, and V. Stuphorn. *The Journal of Neuroscience*, 27(44):11860–11864, 2007.

